# Selecting Landmark Points for Sparse Manifold Learning

**J. G. Silva**
ISEL/ISR
R. Conselheiro Emidio Navarro
1950.062 Lisbon, Portugal
jgs@isel.ipl.pt

**J. S. Marques**
IST/ISR
Av. Rovisco Pais
1949-001 Lisbon, Portugal
jsm@isr.ist.utl.pt

**J. M. Lemos**
INESC-ID/IST
R. Alves Redol, 9
1000-029 Lisbon, Portugal
jlml@inesc-id.pt

## Abstract

There has been a surge of interest in learning non-linear manifold models to approximate high-dimensional data. Both for computational complexity reasons and for generalization capability, sparsity is a desired feature in such models. This usually means dimensionality reduction, which naturally implies estimating the intrinsic dimension, but it can also mean selecting a subset of the data to use as landmarks, which is especially important because many existing algorithms have quadratic complexity in the number of observations. This paper presents an algorithm for selecting landmarks, based on LASSO regression, which is well known to favor sparse approximations because it uses regularization with an $l_1$ norm. As an added benefit, a continuous manifold parameterization, based on the landmarks, is also found. Experimental results with synthetic and real data illustrate the algorithm.

## 1  Introduction

The recent interest in manifold learning algorithms is due, in part, to the multiplication of very large datasets of high-dimensional data from numerous disciplines of science, from signal processing to bioinformatics [6].

As an example, consider a video sequence such as the one in Figure 1. In the absence of features like contour points or wavelet coefficients, each image of size $71 \times 71$ pixels is a point in a space of dimension equal to the number of pixels, $71 \times 71 = 5041$. The observation space is, therefore, $\mathbb{R}^{5041}$. More generally, each observation is a vector $\mathbf{y} \in \mathbb{R}^m$ where $m$ may be very large.

A reasonable assumption, when facing an observation space of possibly tens of thousands of dimensions, is that the data are not dense in such a space, because several of the mea-

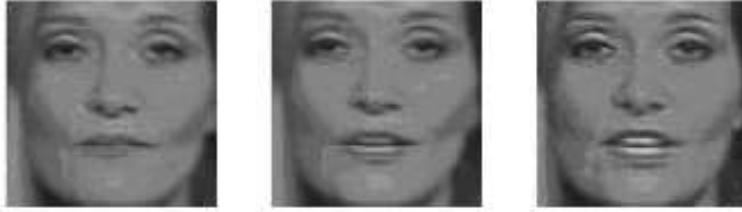

Figure 1: Example of a high-dimensional dataset: each image of size $71 \times 71$ pixels is a point in $\mathbb{R}^{5041}$.

sured variables must be dependent. In fact, in many problems of interest, there are only a few free parameters, which are embedded in the observed variables, frequently in a non-linear way. Assuming that the number of free parameters remains the same throughout the observations, and also assuming smooth variation of the parameters, one is in fact dealing with geometric restrictions which can be well modelled as a manifold.

Therefore, the data must lie on, or near (accounting for noise) a manifold embedded in observation, or ambient space. Learning this manifold is a natural approach to the problem of modelling the data, since, besides computational issues, sparse models tend to have better generalization capability. In order to achieve sparsity, considerable effort has been devoted to reducing the dimensionality of the data by some form of non-linear projection. Several algorithms ([10], [8], [3]) have emerged in recent years that follow this approach, which is closely related to the problem of feature extraction. In contrast, the problem of finding a relevant subset of the observations has received less attention.

It should be noted that the complexity of most existing algorithms is, in general, dependent not only on the dimensionality but also on the number of observations. An important example is the ISOMAP [10], where the computational cost is quadratic in the number of points, which has motivated the L-ISOMAP variant [3] which uses a randomly chosen subset of the points as *landmarks* (L is for Landmark).

The proposed algorithm uses, instead, a principled approach to select the landmarks, based on the solutions of a regression problem minimizing a regularized cost functional. When the regularization term is based on the $l_1$ norm, the solution tends to be sparse. This is the motivation for using the Least Absolute value Subset Selection Operator (LASSO) [5].

Finding the LASSO solutions used to require solving a quadratic programming problem, until the development of the Least Angle Regression (LARS[1]) procedure [4], which is much faster (the cost is equivalent to that of ordinary least squares) and not only gives the LASSO solutions but also provides an estimator of the risk as a function of the regularization tuning parameter. This means that the correct amount of regularization can be automatically found.

In the specific context of selecting landmarks for manifold learning, with some care in the LASSO problem formulation, one is able to avoid a difficult problem of sparse regression with Multiple Measurement Vectors (MMV), which has received considerable interest in its own right [2].

The idea is to use local information, found by local PCA as usual, and preserve the smooth variation of the tangent subspace over a larger scale, taking advantage of any known embedding. This is a natural extension of the Tangent Bundle Approximation (TBA) algorithm, proposed in [9], since the principal angles, which TBA computes anyway, are readily avail-

able and appropriate for this purpose. Nevertheless, the method proposed here is independent of TBA and could, for instance, be plugged into a global procedure like L-ISOMAP.

The algorithm avoids costly global computations, that is, it doesn't attempt to preserve geodesic distances between faraway points, and yet, unlike most local algorithms, it is explicitly designed to be sparse while retaining generalization ability.

The remainder of this introduction formulates the problem and establishes the notation. The selection procedure itself is covered in section 2, while also providing a quick overview of the LASSO and LARS methods. Results are presented in section 3 and then discussed in section 4.

## 1.1 Problem formulation

The problem can be formulated as following: given $N$ vectors $\mathbf{y} \in \mathbb{R}^m$, suppose that the $\mathbf{y}$ can be approximated by a differentiable $n$-manifold $\mathcal{M}$ embedded in $\mathbb{R}^m$. This means that $\mathcal{M}$ can be *charted* through one or more invertible and differentiable mappings of the type

$$g_i(\mathbf{y}) = \mathbf{x} \tag{1}$$

to vectors $\mathbf{x} \in \mathbb{R}^n$ so that open sets $\mathcal{P}_i \subset \mathcal{M}$, called *patches*, whose union covers $\mathcal{M}$, are diffeomorphically mapped onto other open sets $\mathcal{U}_i \subset \mathbb{R}^n$, called *parametric domains*. $\mathbb{R}^n$ is the lower dimensional parameter space and $n$ is the intrinsic dimension of $\mathcal{M}$. The $g_i$ are called *charts*, and manifolds with complex topology may require several $g_i$. Equivalently, since the charts are invertible, inverse mappings $h_i : \mathbb{R}^n \to \mathbb{R}^m$, called *parameterizations* can be also be found.

Arranging the original data in a matrix $\mathbf{Y} \in \mathbb{R}^{m \times N}$, with the $\mathbf{y}$ as column vectors and assuming, for now, only one mapping $g$, the charting process produces a matrix $\mathbf{X} \in \mathbb{R}^{n \times N}$:

$$\mathbf{Y} = \begin{bmatrix} y_{11} & \cdots & y_{1N} \\ \vdots & \ddots & \vdots \\ y_{m1} & \cdots & y_{mN} \end{bmatrix} \quad \mathbf{X} = \begin{bmatrix} x_{11} & \cdots & x_{1N} \\ \vdots & \ddots & \vdots \\ x_{n1} & \cdots & x_{nN} \end{bmatrix} \tag{2}$$

The $n$ rows of $\mathbf{X}$ are sometimes called *features* or *latent variables*. It is often intended in manifold learning to estimate the correct intrinsic dimension, $n$, as well as the chart $g$ or at least a column-to-column mapping from $\mathbf{Y}$ to $\mathbf{X}$. In the present case, this mapping will be assumed known, and so will $n$.

What is intended is to select a subset of the *columns* of $\mathbf{X}$ (or of $\mathbf{Y}$, since the mapping between them is known) to use as landmarks, while retaining enough information about $g$, resulting in a reduced $n \times N'$ matrix with $N' < N$. $N'$ is the number of landmarks, and should also be automatically determined.

Preserving $g$ is equivalent to preserving its inverse mapping, the parameterization $h$, which is more practical because it allows the following generative model:

$$\mathbf{y} = h(\mathbf{x}) + \boldsymbol{\eta} \tag{3}$$

in which $\boldsymbol{\eta}$ is zero mean Gaussian observation noise. How to find the fewest possible landmarks so that $h$ can still be well approximated?

## 2 Landmark selection

### 2.1 Linear regression model

To solve the problem, it is proposed to start by converting the non-linear regression in (3) to a *linear* regression by offloading the non-linearity onto a kernel, as described in numerous works, such as [7]. Since there are $N$ columns in $\mathbf{X}$ to start with, let $\mathbf{K}$ be a square, $N \times N$, symmetric semidefinite positive matrix such that

$$
\begin{aligned}
\mathbf{K} &= \{k_{ij}\} \\
k_{ij} &= K(\mathbf{x}_i, \mathbf{x}_j) \\
K(\mathbf{x}, \mathbf{x}_j) &= \exp(-\frac{\|\mathbf{x} - \mathbf{x}_j\|^2}{2\sigma_K^2}).
\end{aligned}
\tag{4}
$$

The function $K$ can be readily recognized as a Gaussian kernel. This allows the reformulation, in matrix form, of (3) as

$$
\mathbf{Y}^T = \mathbf{K}\mathbf{B} + \mathbf{E}
\tag{5}
$$

,

where $\mathbf{B}, \mathbf{E} \in \mathbb{R}^{N \times m}$ and each line of $\mathbf{E}$ is a realization of $\boldsymbol{\eta}$ above. Still, it is difficult to proceed directly from (5), because neither the response, $\mathbf{Y}^T$, nor the regression parameters, $\mathbf{B}$, are column vectors. This leads to a Multiple Measurement Vectors (MMV) problem, and while there is nothing to prevent solving it separately for each column, this makes it harder to impose sparsity in all columns *simultaneously*. Two alternative approaches present themselves at this point:

- Solve a sparse regression problem for each column of $\mathbf{Y}^T$ (and the corresponding column of $\mathbf{B}$), find a way to force several *lines* of $\mathbf{B}$ to zero.
- Re-formulate (5) is a way that turns it to a single measurement value problem.

The second approach is better studied, and it will be the one followed here. Since the parameterization $h$ is known and must be, at the very least, bijective and continuous, then it must preserve the smoothness of quantities like the geodesic distance and the principal angles. Therefore, it is proposed to re-formulate (5) as

$$
\boldsymbol{\theta} = \mathbf{K}\boldsymbol{\beta} + \boldsymbol{\epsilon}
\tag{6}
$$

where the new response, $\boldsymbol{\theta} \in \mathbb{R}^N$, as well as $\boldsymbol{\beta} \in \mathbb{R}^N$ and $\boldsymbol{\epsilon} \in \mathbb{R}^N$ are now column vectors, allowing the use of known subset selection procedures.

The elements of $\boldsymbol{\theta}$ can be, for example, the geodesic distances to the $\mathbf{y}_\mu = h(\mathbf{x}_\mu)$ observation corresponding to the mean, $\mathbf{x}_\mu$ of the columns of $\mathbf{X}$. This would be a possibility if an algorithm like ISOMAP were used to find the chart from $\mathbf{Y}$ to $\mathbf{X}$. However, since the whole point of using landmarks is to know them beforehand, so as to avoid having to compute $N \times N$ geodesic distances, this is not the most interesting alternative.

A better way is to use a computationally lighter quantity like the maximum principal angle between the tangent subspace at $\mathbf{y}_\mu$, $T_{\mathbf{y}_\mu}(\mathcal{M})$, and the tangent subspaces at all other $\mathbf{y}$.

Given a point $\mathbf{y}_0$ and its $k$ nearest neighbors, finding the tangent subspace can be done by local PCA. The sample covariance matrix $\mathbf{S}$ can be decomposed as

$$\mathbf{S} \;=\; \frac{1}{k}\sum_{i=0}^{k}(\mathbf{y}_i - \mathbf{y}_0)(\mathbf{y}_i - \mathbf{y}_0)^T \qquad (7)$$

$$\mathbf{S} \;=\; \mathbf{V}\mathbf{D}\mathbf{V}^T \qquad (8)$$

where the columns of $\mathbf{V}$ are the eigenvectors $\mathbf{v}_i$ and $\mathbf{D}$ is a diagonal matrix containing the eigenvalues $\lambda_i$, in descending order. The eigenvectors form an orthonormal basis aligned with the principal directions of the data. They can be divided in two groups: tangent and normal vectors, spanning the tangent and normal subspaces, with dimensions $n$ and $m - n$, respectively. Note that $m - n$ is the *codimension* of the manifold. The tangent subspaces are spanned from the $n$ most important eigenvectors. The principal angles between two different tangent subspaces at different points $\mathbf{y}_0$ can be determined from the column spaces of the corresponding matrices $\mathbf{V}$.

An in-depth description of the principal angles, as well as efficient algorithms to compute them, can be found, for instance, in [1]. Note that, should the $T_\mathbf{y}(\mathcal{M})$ be already available from the eigenvectors found during some local PCA analysis, e. g., during estimation of the intrinsic dimension, there would be little extra computational burden. An example is [9], where the principal angles already are an integral part of the procedure - namely for partitioning the manifold into patches.

Thus, it is proposed to use $\theta_j$ equal to the maximum principal angle between $T_{\mathbf{y}_\mu}(\mathcal{M})$ and $T_{\mathbf{y}_j}(\mathcal{M})$, where $\mathbf{y}_j$ is the $j$-th column of $\mathbf{Y}$. It remains to be explained how to achieve a sparse solution to (6).

## 2.2 Sparsity with LASSO and LARS

The idea is to find an estimate $\hat{\boldsymbol{\beta}}$ that minimizes the functional

$$E = \|\boldsymbol{\theta} - \mathbf{K}\hat{\boldsymbol{\beta}}\|^2 + \gamma\|\hat{\boldsymbol{\beta}}\|_q^q. \qquad (9)$$

Here, $\|\hat{\boldsymbol{\beta}}\|_q$ denotes the $l_q$ norm of $\hat{\boldsymbol{\beta}}$, i. e. $\sqrt[q]{\sum_{i=1}^{m}|\hat{\beta}_i|^q}$, and $\gamma$ is a tuning parameter that controls the amount of regularization. For the most sparseness, the ideal value of $q$ would be zero. However, minimizing $E$ with the $l_0$ norm is, in general, prohibitive in computational terms. A sub-optimal strategy is to use $q = 1$ instead. This is the usual formulation of a LASSO regression problem. While minimization of (9) can be done using quadratic programming, the recent development of the LARS method has made this unnecessary. For a detailed description of LARS and its relationship with the LASSO, *vide* [4].

Very briefly, LARS starts with $\hat{\boldsymbol{\beta}} = \mathbf{0}$ and adds covariates (the columns of $\mathbf{K}$) to the model according to their correlation with the prediction error vector, $\boldsymbol{\theta} - \mathbf{K}\hat{\boldsymbol{\beta}}$, setting the corresponding $\hat{\beta}_j$ to a value such that another covariate becomes equally correlated with the error and is, itself, added to the model - it becomes *active*. LARS then proceeds in a direction equiangular to all the active $\hat{\beta}_j$ and the process is repeated until all covariates have been added. There are a total of $m$ steps, each of which adds a new $\hat{\beta}_j$, making it non-zero. With slight modifications, these steps correspond to a sampling of the tuning parameter $\gamma$ in (9) under LASSO. Moreover, [4] shows that the risk, as a function of the number, $p$, of non-zero $\hat{\beta}_j$, can be estimated (under mild assumptions) as

$$R(\hat{\boldsymbol{\beta}}_p) = \|\boldsymbol{\theta} - \mathbf{K}\hat{\boldsymbol{\beta}}_p\|^2/\bar{\sigma}^2 - m + 2p \qquad (10)$$

where $\bar{\sigma}^2$ can be found from the unconstrained least squares solution of (6). Computing $R(\hat{\boldsymbol{\beta}}_p)$ requires no more than the $\hat{\boldsymbol{\beta}}_p$ themselves, which are already provided by LARS anyway.

### 2.3 Landmarks and parameterization of the manifold

The landmarks are the columns $\mathbf{x}_j$ of $\mathbf{X}$ (or of $\mathbf{Y}$) with the same indexes $j$ as the non-zero elements of $\boldsymbol{\beta}_p$, where

$$p = \arg \min_p R(\boldsymbol{\beta}_p). \tag{11}$$

There are $N' = p$ landmarks, because there are $p$ non-zero elements in $\boldsymbol{\beta}_p$. This criterion ensures that the landmarks are the kernel centers that minimize the risk of the regression in (6).

As an interesting byproduct, regardless of whether $h$ was a continuous or point-to-point mapping to begin with, it is now also possible to obtain a new, continuous parameterization $h_{\mathbf{B},\mathbf{X}'}$ by solving a reduced version of (5):

$$\mathbf{Y}^T = \mathbf{B}\mathbf{K}' + \mathbf{E} \tag{12}$$

where $\mathbf{K}'$ only has $N'$ columns, with the same indexes as $\mathbf{X}'$. In fact, $\mathbf{K}' \in \mathbb{R}^{N \times N'}$ is no longer square. Also, now $\mathbf{B} \in \mathbb{R}^{N' \times m}$. The new, smaller regression (12) can be solved separately for each column of $\mathbf{Y}^T$ and $\mathbf{B}$ by *unconstrained* least squares. For a new feature vector, $\mathbf{x}$, in the parametric domain, a new vector $\mathbf{y} \in \mathcal{M}$ in observation space can be synthesized by

$$\begin{aligned}
\mathbf{y} = h_{\mathbf{B},\mathbf{X}'}(\mathbf{x}) &= [y_1(\mathbf{x}) \ldots y_m(\mathbf{x})]^T \\
y_j(\mathbf{x}) &= \sum_{\mathbf{x}_i \in \mathbf{X}'} b_{ij} K(\mathbf{x}_i, \mathbf{x})
\end{aligned} \tag{13}$$

where the $\{b_{ij}\}$ are the elements of $\mathbf{B}$.

## 3 Results

The algorithm has been tested in two synthetic datasets: the traditional synthetic "swiss roll" and a sphere, both with $1000$ points embedded in $\mathbb{R}^{10}$, with a small amount of isotropic Gaussian noise ($\sigma_\mathbf{y} = 0.01$) added in all dimensions, as shown in Figure 2. These manifolds have intrinsic dimension $n = 2$. A global embedding for the swiss roll was found by ISOMAP, using $k = 8$. On the other hand, TBA was used for the sphere, resulting in multiple patches and charts - a necessity, because otherwise the sphere's topology would make ISOMAP fail. Therefore, in the sphere, each patch has its own landmark points, and the manifold require the union of all such points. All are shown in Figure 2, as selected by our procedure.

Additionally, a real dataset was used: images from the video sequence shown above in Figure 1. This example is known [9] to be reasonably well modelled by as few as 2 free parameters.

The sequence contains $N = 194$ frames with $m = 5041$ pixels. A first step was to perform global PCA in order to discard irrelevant dimensions. Since it obviously isn't possible

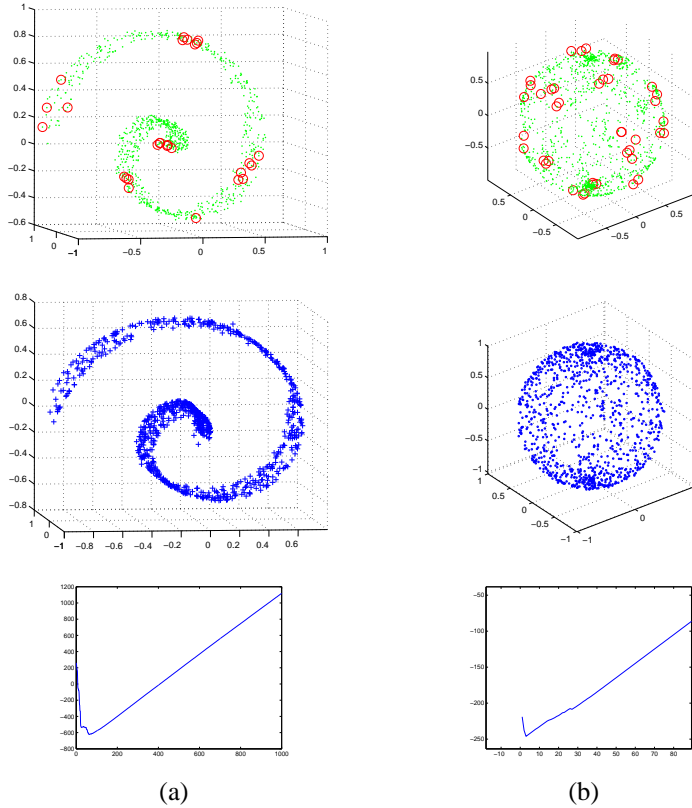

(a)                                    (b)

Figure 2: *Above*: landmarks; *Middle*: interpolated points using $h_{\mathbf{B},\mathbf{X}'}$; *Below*: risk esti-
mates. For the sphere, the risk plot is for the largest patch. Total landmarks, $N' = 27$ for
the swiss roll, $42$ for the sphere.

to compute a covariance matrix of size $5000 \times 5000$ from $194$ samples, the problem was
transposed, leading to the computation of the eigenvectors of a $N \times N$ covariance, from
which the first $N - 1$ eigenvectors of the non-transposed problem can easily be found [11].
This resulted in an estimated 15 globally significant principal directions, on which the data
were projected.

After this pre-processing, the effective values of $m$ and $N$ were, respectively, 15 and 194.
An embedding was found using TBA with 2 features (ISOMAP would have worked as
well). The results obtained for this case are shown in Figure 3. Only 4 landmarks were
needed, and they correspond to very distinct face expressions.

## 4   Discussion

A new approach for selecting landmarks in manifold learning, based on LASSO and LARS
regression, has been presented. The proposed algorithm finds geometrically meaningful
landmarks and successfully circumvents a difficult MMV problem, by using the intuition
that, since the variation of the maximum principal angle is a measure of curvature, the
points that are important in preserving it should also be important in preserving the overall
manifold geometry. Also, a continuous manifold parameterization is given with very little

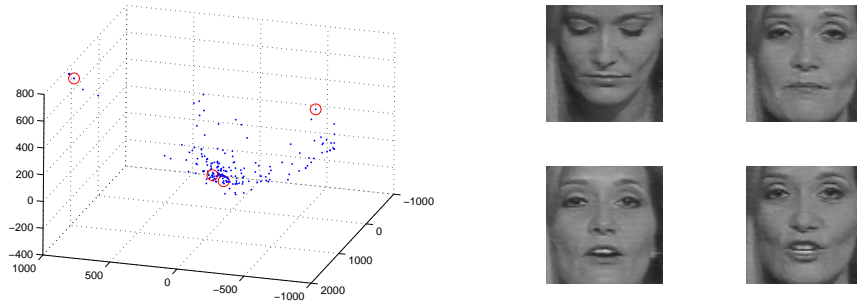

Figure 3: Landmarks for the video sequence: $N' = 4$, marked over a scatter plot of the first 3 eigen-coordinates. The corresponding pictures are also shown.

additional computational cost.

The entire procedure avoids expensive, quadratic programming computations - its complexity is dominated by the LARS step, which has the same cost as a least squares fit [4]. The proposed approach has been validated with experiments on synthetic and real datasets.

## Acknowledgments

This work was partially supported by FCT POCTI, under project 37844.

## Footnotes

[1]The S in LARS stands for Stagewise and LA*SS*O, an allusion to the relationship between the three algorithms.

## References

[1] A. Bjorck and G. H. Golub. Numerical methods for computing angles between linear subspaces. *Mathematical Computation*, 27, 1973.

[2] J. Chen and X. Huo. Sparse representation for multiple measurement vectors (mmv) in an over-complete dictionary. *ICASSP*, 2005.

[3] V. de Silva and J. B. Tenenbaum. Global versus local methods in nonlinear dimensionality reduction. *NIPS*, 15, 2002.

[4] B. Efron, T. Hastie, I. Johnstone, and R. Tibshirani. Least angle regression. *Annals of Statistics*, 2003.

[5] T. Hastie, R. Tibshirani, and J. H. Friedman. *The Elements of Statistical Learning*. Springer, 2001.

[6] H. Lädesmäki, O. Yli-Harja, W. Zhang, and I. Shmulevich. Intrinsic dimensionality in gene expression analysis. *GENSIPS*, 2005.

[7] T. Poggio and S. Smale. The mathematics of learning: Dealing with data. *Notices of the American Mathematical Society*, 2003.

[8] S. T. Roweis and L. K. Saul. Nonlinear dimensionality reduction by locally linear embedding. *Science*, 290:2323–2326, 2000.

[9] J. Silva, J. Marques, and J. M. Lemos. Non-linear dimension reduction with tangent bundle approximation. *ICASSP*, 2005.

[10] J. B. Tenenbaum, V. de Silva, and J. C. Langford. A global geometric framework for nonlinear dimensionality reduction. *Science*, 290:2319–2323, 2000.

[11] M. Turk and A. Pentland. Eigenfaces for recognition. *Journal of Cognitive Neuroscience*, 3:71–86, 1991.
